# Dendritic compartmentalization could underlie competition and attentional biasing of simultaneous visual stimuli

**Kevin A. Archie**
Neuroscience Program
University of Southern California
Los Angeles, CA 90089-2520

**Bartlett W. Mel**
Department of Biomedical Engineering
University of Southern California
Los Angeles, CA 90089-1451

## Abstract

Neurons in area V4 have relatively large receptive fields (RFs), so multiple visual features are simultaneously "seen" by these cells. Recordings from single V4 neurons suggest that simultaneously presented stimuli compete to set the output firing rate, and that attention acts to isolate individual features by biasing the competition in favor of the attended object. We propose that both stimulus competition and attentional biasing arise from the spatial segregation of afferent synapses onto different regions of the excitable dendritic tree of V4 neurons. The pattern of feedforward, stimulus-driven inputs follows from a Hebbian rule: excitatory afferents with similar RFs tend to group together on the dendritic tree, avoiding randomly located inhibitory inputs with similar RFs. The same principle guides the formation of inputs that mediate attentional modulation. Using both biophysically detailed compartmental models and simplified models of computation in single neurons, we demonstrate that such an architecture could account for the response properties and attentional modulation of V4 neurons. Our results suggest an important role for nonlinear dendritic conductances in extrastriate cortical processing.

## 1 Introduction

Neurons in higher regions of visual cortex have relatively large receptive fields (RFs): for example, neurons representing the central visual field in macaque area V4 have RFs up to 5° across (Desimone & Schein, 1987). Such large RFs often contain multiple potentially significant features in a single image, leading to the question: How can these neurons extract information about individual objects? Moran and Desimone (1985) showed that when multiple stimuli are present within the RF of a V4 neuron, attention effectively reduces the RF extent of the cell, so that only the attended feature contributes to its output. Desimone (1992) noted that one way this modulation could be performed is to assign input from each RF subregion to a single dendritic branch of the V4 neuron; modulatory inhibition could then "turn off" branches, so that subregions of the RF could be independently gated.

Recent experiments have revealed a more subtle picture regarding both the interactions between simultaneously presented stimuli and the effects of attentional modulation. Record-

ings from individual V4 neurons have shown that simultanously presented stimuli compete to set the output firing rate (Luck, Chelazzi, Hillyard, & Desimone, 1997; Reynolds, Chelazzi, & Desimone, 1999). For example, consider a cell for which stimulus $S$, presented by itself, produces a strong response consisting of $s$ spikes, and stimulus $W$ produces a weak response of $w$ spikes. Presenting the two stimuli $S$ and $W$ together generally produces an output less than $s$ but more than $w$. Note that the "weak" stimulus $W$ is excitatory for the cell when presented alone, since it increases the response from 0 to $w$, but effectively inhibitory when presented together with "strong" stimulus $S$. Attention serves to bias the competition, so that attending to $S$ would increase the output of the V4 cell (moving it closer to $s$), while attending to $W$ would decrease the output (moving it closer to $w$). To describe their results, Reynolds et al. (1999) proposed a mathematical model in which individual stimuli both excite and inhibit the V4 neuron. The sum of excitatory and inhibitory input is acted on by divisive normalization proportional to the total strength of input to produce a competitive interaction between simultaneous stimuli. Attention is then implemented as a multiplicative gain on both excitatory and inhibitory input arising from the attended stimulus.

In previous work using biophysically detailed compartmental models of neurons with active dendrites, we observed that increasing the stimulus contrast produced a multiplicative scaling of the tuning curve of a complex cell (Archie & Mel, 2000, Fig. 6g), suggesting an implicit normalization. In the present work, we test the following hypotheses: (1) segregation of input onto different branches of an excitable dendritic tree could produce competitive interactions between simultaneously presented stimuli, and (2) modulatory synapses on active dendrites could be a general mechanism for multiplicative modulation of inputs.

## 2  Methods

We used both biophysically detailed compartmental models and a simplified model of a single cortical neuron to test whether competition and attentional biasing could arise from interactions between excitatory and inhibitory inputs in a nonlinear dendritic tree. An overview of the input segregation common to both classes of model is shown in Fig. 1.

**Biophysically detailed compartmental model.**  The detailed model included 4 layers of processing: (1) an LGN cell layer with center-surround RFs; (2) a virtual layer of simple-cell-like subunits which were drawn from elongated rows of ON- and OFF-center LGN cells — virtual in that the subunit computations were actually carried out in the dendrites of the overlying complex cells, following Mel, Ruderman, and Archie (1998) and Archie and Mel (2000); (3) an $8 \times 8$ grid of complex cells, each of which contained 4 subunits with progressively shifted positions/phases; and (4) a single V4 cell in the top layer, which received input from the complex cell layer. Layers 3 and 4 are shown in Fig. 2.

The LGN was modeled as 4 arrays (ON- and OFF-center, left and right eye) of difference-of-Gaussian spatial filters, as described in Archie and Mel (2000). Responses of the cortical cells were calculated using the NEURON simulation environment (Hines & Carnevale, 1997). Complex cells contained 4 basal branches, each 1 $\mu$m in diameter and 150 $\mu$m long; one apical branch 5 $\mu$m in diameter and 250 $\mu$m long; a spherical soma 20 $\mu$m in diameter; and an axon 0.5 $\mu$m in diameter and 1000 $\mu$m long with an initial segment 1 $\mu$m in diameter and 20 $\mu$m long. Hodgkin-Huxley-style Na$^+$ and K$^+$ conductances were present in the membrane of the entire cell, with 10-fold higher density in the axon ($\bar{g}_{Na} = 0.120$ S/cm$^2$, $\bar{g}_K = 0.100$ S/cm$^2$) than in the soma and dendrites ($\bar{g}_{Na} = 0.012$ S/cm$^2$, $\bar{g}_K = 0.010$ S/cm$^2$). The V4 cell was modeled with the same parameters as the complex cells, but with 8 basal branches instead of 4.

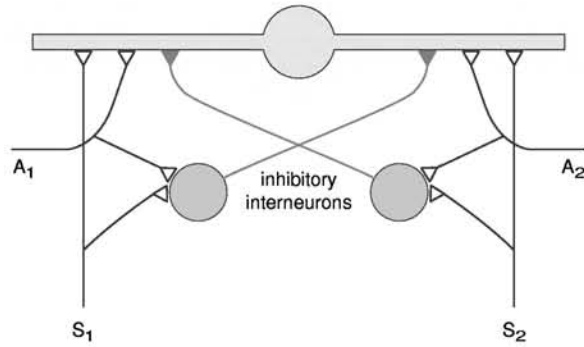

Figure 1: Segregation of excitatory and inhibitory inputs. Two sources of stimulus-driven input are shown, $S_1$ and $S_2$, each corresponding to an independently attendable subregion of the RF of the V4 cell. Note that each source of stimulus-driven input makes both excitatory projections to a specific branch on the V4 cell, and inhibitory projections (through an interneuron) to other branches. Similarly, the modulatory inputs $A_1$ and $A_2$ each direct attention to a particular branch; for example, $A_1$ adds excitatory modulation to the branch corresponding to the $S_1$ RF subregion and (indirectly) inhibitory modulation to other branches.

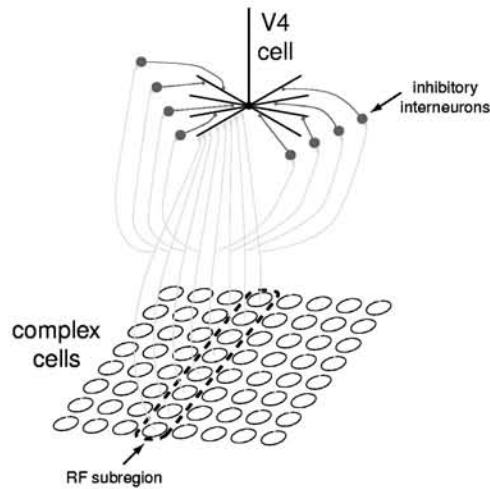

Figure 2: Design of the biophysically detailed model. Complex cells were arranged in a grid with overlapping RFs and similar orientation preferences. Each vertical stripe of cells formed an RF subregion to which attention could be directed. Each complex cell within a subregion formed one excitatory and one (indirect) inhibitory connection onto the V4 cell. Synapse locations were assigned at random within a given V4 branch. All excitatory connections for a given subregion were targeted to a single branch, while the corresponding inhibitory synapses were distributed across all of the other branches of the cell. Attentional modulatory and background synapses, both described in the text, are not shown.

Excitatory synapses were modeled as having both voltage-dependent (NMDA) and voltage-independent (AMPA/kainate) components, while inhibitory synapses were fast and voltage-independent (GABA-A). All synapses were modeled using the kinetic scheme of Destexhe, Mainen, and Sejnowski (1994), with peak conductance values scaled inversely by the local input resistance to reduce the dependence of local EPSP size on synapse location.

The complex cells received input from the LGN, using the spatial arrangement of excitatory and inhibitory inputs described in Archie and Mel (2000), with inhibitory inputs distributed throughout the input branches (rather than, e.g., being restricted to the proximal part of the branch). We have previously shown that this arrangement of inputs produces phase- and contrast-invariant tuning to stimulus orientation, similar to that seen in cortical complex cells. All 64 complex cells had the same preferred orientation, which we will for convenience call vertical. For each stimulus image, each complex cell was simulated for 1 second and the resulting firing rate was used to set the activity level of one excitatory and one inhibitory synapse onto the V4 cell. The inhibition was assumed to emanate from an implicit inhibitory interneuron in V4. The stimulus-driven inputs to the V4 neuron were modeled as Poisson trains whose mean rate was set to the corresponding complex cell firing rate for excitatory synapses, and 1.3 times the corresponding complex cell firing rate for inhibitory synapses. The inputs were arranged on the V4 cell so that all of the complex cells with RFs distributed along a vertical stripe of the V4 RF (i.e., aligned with the preferred orientation of the complex cells) formed one subregion and made their excitatory projections to a single branch (Fig. 2). The inhibitory synapse from each complex cell was placed on a different branch than the corresponding excitatory synapse, with the specific location chosen at random.

Attention was implemented by placing two modulatory synapses on each branch, one excitatory and one inhibitory. In the absence of attention, all modulatory synapses had a mean event rate of 0.1 Hz. Attention was directed to a particular subregion by increasing the firing rate of the excitatory modulation on the corresponding branch to 100 Hz, and increasing the inhibitory modulation on all other branches to 67 Hz. Each branch of the V4 cell also received a single excitatory synapse with mean firing rate 25 Hz, representing background (non-stimulus driven) input from the cortical network. These synapses provided most of the input needed for the cell to fire action potentials, while the stimulus-driven inputs modulated the firing rate.

The rationale for the spatial arrangement of synapses was that coaligned complex cells with overlapping RFs would have correlated responses over the ensemble of images seen during early postnatal development, and would thus tend to congregate within the same dendritic subunits according to Hebbian developmental principles. Similarly, excitatory synapses would tend to avoid inhibitory synapses driven by the same stimuli, since if the two are near each other on the dendrite, the efficacy of the excitation is systematically reduced by the corresponding inhibition.

**Sum of squared filters model.** We have previously proposed that an individual cortical pyramidal neuron may carry out high-order computations that roughly fit the form of an energy model, i.e., a sum of half-squared linear filters, with electrotonically isolated regions of the dendritic tree performing the quadratic subunit computations. Only excitatory inputs were previously considered, leaving open the question of how inhibition might fit in such a model. An obvious implementation of inhibition is to simply subtract the mean firing rates of inhibitory inputs, just as excitatory inputs are added. The sum-of-squares model thus has the form: $f(\mathbf{x}) = \sum_j ((\sum_{i \in \mathcal{B}_j} w_i x_i)^+)^2$, where $y^+$ denotes $y$ if $y \geq 0$, 0 otherwise; $\mathcal{B}_j$ is the set of inputs $i$ that project to branch $j$; and $w_i$ is $+1$ if input $i$ is excitatory, $-1$ if inhibitory. We considered both a "paper-and-pencil" model, in which we hand-selected input values for each stimulus with an eye towards ease of interpretation; and also a model in which the tabulated complex cell output rates (from layer 3 of the detailed model) were

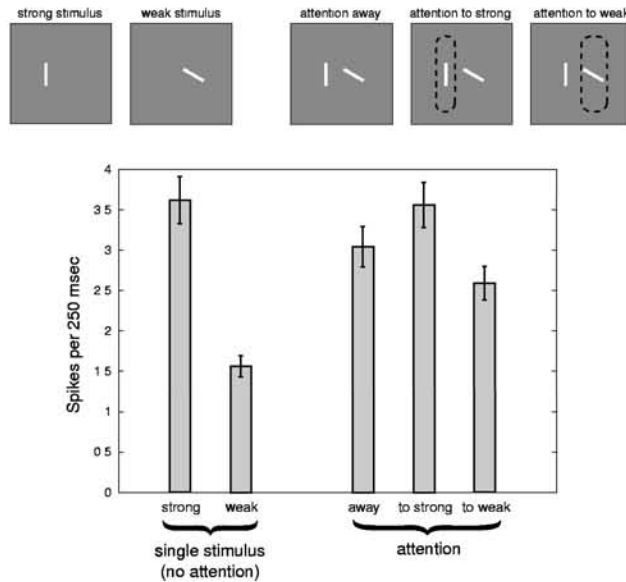

Figure 3: Results from the biophysically detailed model. In the top row, strong and weak visual stimuli are shown at left, and combined stimuli in three attentional conditions are indicated at right. Bar graph shows response of simulated V4 cell under each of these 5 conditions, averaged over 192 runs. Combined stimulus in the absence of attention yields output between the responses to either stimulus alone. Attention to either the strong or the weak stimulus pushes the cell's response toward the individual response for that stimulus.

used as input.

# 3 Results

**Detailed model.** A strong stimulus (a vertical bar) and a weak stimulus (a bar of the same length, turned 60° from vertical) were selected. Figure 3 shows the stimulus images and simulated V4 cell response for each stimulus alone and for the combined stimulus in various attentional states. In the absence of attention within the receptive field (attention away), the response of the cell to the combined image lay between the responses to the strong image alone or the weak image alone. This intermediate response is consistent with the responses of many V4 cells under similar conditions, and is the result of the competition between excitatory and inhibitory inputs: because of the spatial segregation, inhibitory synapses driven by one stimulus selectively undermine the effectiveness of excitation due to the other.

This competition between stimuli was also biased by attentional modulation (Fig. 3). Attending to the strong stimulus elevated the response to the combined image compared to the condition where attention was directed away, thus bringing the response closer to the response to the strong stimulus alone. Similarly, attention to the weak stimulus lowered the response to the combined stimulus.

**Sum of squared filters.** We used a 4-subunit sum-of-squares model for illustrative purposes. A stimulus in this model is a 4-dimensional vector, with each component representing the total input (excitatory positive, inhibitory negative) to a single subunit. Most

stimuli tested had equal excitatory and inhibitory influence, so that the sum of the components was zero, and had excitatory influence confined to one subunit (i.e., the features were small compared to the entire V4 RF). One example set of stimulus vectors follows, with $\bar{\mathbf{x}}$ indicating that stimulus $\mathbf{x}$ is attended (implemented by adding a modulatory value of $+1$ to the attended branch, and $-1$ to all others):

$$
\begin{array}{rccll}
\mathbf{s} = & [5, -2, -1, -2] & \longrightarrow & 25 + 0 + 0 + 0 & = 25 \\
\mathbf{w} = & [-1, -1, -1, 3] & \longrightarrow & 0 + 0 + 0 + 9 & = 9 \\
\mathbf{s} + \mathbf{w} = & [4, -3, -2, 1] & \longrightarrow & 16 + 0 + 0 + 1 & = 17 \\
\bar{\mathbf{s}} + \mathbf{w} = & [5, -4, -3, 0] & \longrightarrow & 25 + 0 + 0 + 0 & = 25 \\
\mathbf{s} + \bar{\mathbf{w}} = & [3, -4, -3, 2] & \longrightarrow & 9 + 0 + 0 + 4 & = 13
\end{array}
$$

This simple model gave qualitiatively correct results. Some stimulus combinations we considered gave results inconsistent with the biased-competition model — e.g., the above situation with $\mathbf{w} = [-1, -1, 3, -1]$. The most common type of failure was that attending to the strong stimulus in the combined image led to a larger response than that produced by the strong stimulus alone. We also saw this happen for certain parameter sets in the biophysically detailed model, as described below; a similar result is seen in some of the data of Reynolds et al. (1999). Nonetheless, this simple model gives qualitatively correct results for a surprisingly large set of input combinations.

When the complex-cell output from the detailed model was used as input to a sum-of-squared-filters model with 8 subunits, results qualitatively similar to the detailed simulation results were obtained. For the stimuli shown in Fig. 3, the following results were seen (all responses in arbitrary units): with no attention, strong: 109, weak: 2.57, combined: 84.3; combined, attention to strong: 106; to weak: 80. This simplified model, like the biophysically detailed model, is rather sensitive to the values used for the modulatory inputs: with slightly different values, for example, attending to the strong stimulus makes the response to the combined image higher than the response to the strong stimulus alone. In continuing studies, we are working to determine whether this parameter sensitivity is a general feature of such models.

## 4 Discussion

A variety of previous models for attention have considered how the RF of cortical neurons can be dynamically modulated (Olshausen, Anderson, & Van Essen, 1993; Niebur & Koch, 1994; Salinas & Abbott, 1997; Lee, Itti, Koch, & Braun, 1999). Our model, an extension of the proposal of Desimone (1992), specifies a biophysical mechanism for the multiplicative gain used in previous models (Salinas & Abbott, 1997; Reynolds et al., 1999), and suggests that both the stimulus competition and attentional effects seen in area V4 could be implemented by a straightforward mapping of stimulus-driven and modulatory afferents, both excitatory and inhibitory, onto the dendrites of V4 neurons. The results from the sum-of-squared-filters models demonstrate that even a crude model of computation in single neurons can account for the complicated response properties of V4 neurons, given several quasi-independent nonlinear dendritic subunits and a suitable spatial arrangement of synapses. In continuing work, we are exploring the large space of parameters (e.g., density of various ionic conductances, ratio of inhibition to excitation, strength of modulatory inputs) to determine which aspects of the response properties are fundamental to the model, and which are accidents of the particular parameters chosen. This work should help to identify strong vs. weak experimental predictions regarding the contributions of dendritic subunit computation to the response properties of extrastriate neurons.

## Acknowledgements

Supported by NSF.

# Reference

Archie, K. A., & Mel, B. W. (2000). A model for intradendritic computation of binocular disparity. *Nature Neurosci.*, *3*(1), 54–63.

Connor, C. E., Preddie, D. C., Gallant, J. L., & Essen, D. C. V. (1997). Spatial attention effects in macaque area V4. *J. Neurosci.*, *17*(9), 3201–3214.

Desimone, R., & Schein, S. J. (1987). Visual properties of neurons in area V4 of the macaque: sensitivity to stimulus form. *J. Neurophysiol.*, *57*(3), 835–868.

Desimone, R. (1992). Neural circuits for visual attention in the primate brain. In Carpenter, G. A., & Grossberg, S. (Eds.), *Neural Networks for Vision and Image Processing*, chap. 12, pp. 343–364. MIT Press, Cambridge, MA.

Desimone, R. (1998). Visual attention mediated by biased competition in extrastriate visual cortex. *Phil. Trans. R. Soc. Lond. B*, *353*, 1245–1255.

Destexhe, A., Mainen, Z., & Sejnowski, T. J. (1994). Synthesis of models for excitable membranes, synaptic transmission and neuromodulation using a common kinetic formalism. *J. Comput. Neurosci.*, *1*, 195–230.

Destexhe, A., & Paré, D. (1999). Impact of network activity on the integrative properties of neocortical pyramidal neurons in vivo. *J. Neurophysiol.*, *81*, 1531–1547.

Hines, M. L., & Carnevale, N. T. (1997). The NEURON simulation environment. *Neural Comput.*, *9*, 1179–1209.

Lee, D. K., Itti, L., Koch, C., & Braun, J. (1999). Attention activates winner-take-all competition among visual filters. *Nature Neurosci.*, *2*(4), 375–381.

Luck, S. J., Chelazzi, L., Hillyard, S. A., & Desimone, R. (1997). Neural mechanisms of spatial selective attention in areas V1, V2, and V4 of macaque visual cortex. *J. Neurophysiol.*, *77*, 24–42.

McAdams, C. J., & Maunsell, J. H. R. (1999). Effects of attention on orientation-tuning functions of single neurons in macaque cortical area V4. *J. Neurosci.*, *19*(1), 431–441.

Mel, B. W. (1999). Why have dendrites? A computational perspective. In Stuart, G., Spruston, N., & Häusser, M. (Eds.), *Dendrites*, chap. 11, pp. 271–289. Oxford University Press.

Mel, B. W., Ruderman, D. L., & Archie, K. A. (1998). Translation-invariant orientation tuning in visual "complex" cells could derive from intradendritic computations. *J. Neurosci.*, *18*(11), 4325–4334.

Moran, J., & Desimone, R. (1985). Selective attention gates visual processing in the extrastriate cortex. *Science*, *229*, 782–784.

Motter, B. C. (1993). Focal attention produces spatially selective processing in visual cortical areas V1, V2, and V4 in the presence of competing stimuli. *J. Neurophysiol.*, *70*(3), 909–919.

Niebur, E., & Koch, C. (1994). A model for the neuronal implementation of selective visual attention based on temporal correlation among neurons. *J. Comput. Neurosci.*, *1*, 141–158.

Olshausen, B. A., Anderson, C. H., & Van Essen, D. C. (1993). A neurobiological model of visual attention and invariant pattern recognition based on dynamic routing of information. *J. Neurosci.*, *13*(11), 4700–4719.

Reynolds, J. H., Chelazzi, L., & Desimone, R. (1999). Competitive mechanisms subserve attention in macaque areas V2 and V4. *J. Neurosci.*, *19*(5), 1736–1753.

Salinas, E., & Abbott, L. F. (1997). Invariant visual responses from attentional gain fields. *J. Neurophysiol.*, *77*, 3267–3272.
